# A Minimal Intervention Principle for Coordinated Movement

**Emanuel Todorov**
Department of Cognitive Science
University of California, San Diego
todorov@cogsci.ucsd.edu

**Michael I. Jordan**
Computer Science and Statistics
University of California, Berkeley
jordan@cs.berkeley.edu

## Abstract

Behavioral goals are achieved reliably and repeatedly with movements rarely reproducible in their detail. Here we offer an explanation: we show that not only are variability and goal achievement compatible, but indeed that allowing variability in redundant dimensions is the optimal control strategy in the face of uncertainty. The optimal feedback control laws for typical motor tasks obey a "minimal intervention" principle: deviations from the average trajectory are only corrected when they interfere with the task goals. The resulting behavior exhibits task-constrained variability, as well as synergetic coupling among actuators—which is another unexplained empirical phenomenon.

## 1 Introduction

Both the difficulty and the fascination of the motor coordination problem lie in the apparent conflict between two fundamental properties of the motor system: the ability to accomplish its goal reliably and repeatedly, and the fact that it does so with variable movements [1]. More precisely, trial-to-trial fluctuations in individual degrees of freedom are on average larger than fluctuations in task-relevant movement parameters—motor variability is constrained to a redundant or "uncontrolled" manifold [16] rather than being suppressed altogether. This pattern has now been observed in a long list of behaviors [1, 6, 16, 14]. In concordance with such naturally occurring variability, experimentally induced perturbations [1, 3, 12] are compensated in a way that maintains task performance rather than a specific stereotypical movement pattern.

This body of evidence is fundamentally incompatible with standard models of motor coordination that enforce a strict separation between trajectory planning and trajectory execution [2, 8, 17, 10]. In such serial planning/execution models, the role of the planning stage is to resolve the redundancy inherent in the musculo-skeletal system, by replacing the behavioral goal (achievable via infinitely many movement trajectories) with a specific "desired trajectory." Accurate execution of the desired trajectory guarantees achievement of the goal, and can be implemented with relatively simple trajectory-tracking algorithms. While this approach is computationally viable (and often used in engineering), the numerous observations of task-constrained variability and goal-directed corrections indicate that the online execution mechanisms are able to distinguish, and selectively enforce, the details that are crucial for the achievement of the goal. This would be impossible if the behavioral

goal were replaced with a specific trajectory.

Instead, these observations imply a very different control scheme, one which pursues the behavioral goal more directly. Efforts to delineate such a control scheme have led to the idea of motor synergies, or high-level "control knobs," that have invariant and predictable effects on the task-relevant movement parameters despite variability in individual degrees of freedom [9, 11]. But the computational underpinnings of such an approach—how the synergies appropriate for a given task and plant can be constructed, what control scheme is capable of utilizing them, and why the motor system should prefer such a control scheme in the first place—remain unclear. This general form of hierarchical control implies correlations among the control signals sent to multiple actuators (i.e., synergetic coupling) and a corresponding reduction in control space dimesionality. Such phenonema have indeed been observed [4, 18], but the relationship to the hypothetical functional synergies remains to be established.

In this paper we aim to resolve the apparent conflict at the heart of the motor coordination problem, and clarify the relationship between variability, task goals, and motor synergies. We treat motor coordination within the framework of stochastic optimal control, and postulate that the motor system approximates the *best* possible control scheme for a given task. Such a control scheme will generally take the form of a feedback control law. Whenever the task allows redundant solutions, the initial state of the plant is uncertain, the consequences of the control signals are uncertain, and the movement duration exceeds the shortest sensory-motor delay, optimal performance is achieved by a feedback control law that resolves redundancy moment-by-moment—using all available information to choose the most advantageous course of action under the present circumstances. By postponing all decisions regarding movement details until the last possible moment, this control law takes advantage of the opportunities for more successful task completion that are constantly being created by unpredictable fluctuations away from the average trajectory. Such exploitation of redundancy not only results in higher performance, but also gives rise to task-constrained variability and motor synergies—the phenomena we seek to explain.

The present paper is related to a recent publication targeted at a neuroscience audience [14]. Here we provide a number of technical results missing from [14], and emphasize the aspects of our work that are most likely to be of interest to the computational modeling community.

## 2    The Minimal Intervention principle

Our general explanation of the above phenomena follows from an intuitive property of optimal feedback controllers which we call the "minimal intervention" principle: *deviations from the average trajectory are corrected only when they interfere with task performance.*

If this principle holds, and the noise perturbs the system in all directions, the interplay of the noise and control processes will result in variability which is larger in task-irrelevant directions. At the same time, the fact that certain deviations are not being corrected implies that the corresponding control subspace is not being used—which is the phenomenon typically interpreted as evidence for motor synergies [4, 18].

Why should the minimum intervention principle hold? An optimal feedback controller has nothing to gain from correcting task-irrelevant deviations, because its only concern is task performance and by definition such deviations do not interfere with performance. On the other hand, generating a corrective control signal can be detrimental, because: 1) the noise in the motor system is known to be multiplicative [13] and therefore could increase; 2) the cost being minimized most likely includes a control-dependent effort penalty which could also increase.

We now formalize the notions of "redundancy" and "correction," and show that for a surprisingly general class of systems they are indeed related—as our intuition suggests.

## 2.1 Local analysis of a general class of optimal control problems

Redundancy is not easy to define. Consider the task of reaching, which requires the fingertip to be at a specified target at some point in time $T$. At time $T$, all arm configurations for which the fingertip is at the target are redundant. But at times different from $T$ this geometric approach is insufficient to define redundancy. Therefore we follow a more general approach.

Consider a system with state $\mathbf{x}(t) \in \mathbb{R}^m$, control $\mathbf{u}(t) \in \mathbb{R}^n$, instantaneous scalar cost $\ell(t, \mathbf{x}(t), \mathbf{u}(t)) \geq 0$, and dynamics

$$d\mathbf{x} = \mathbf{f}(t, \mathbf{x}, \mathbf{u}) \, dt + G(t, \mathbf{x}, \mathbf{u}) \, d\varepsilon$$

where $\varepsilon(t) \in \mathbb{R}^k$ is multidimensional standard Brownian motion. Control signals are generated by a feedback control law, which can be any mapping of the form $\mathbf{u}(t) = \boldsymbol{\pi}(t, \mathbf{x}(t))$. The analysis below heavily relies on properties of the optimal cost-to-go function, defined as

$$v^*(t, \mathbf{x}) = \min_{\boldsymbol{\pi}(\cdot, \cdot)} E_{\mathbf{x}(\cdot)} \int_t^T \ell(s, \mathbf{x}(s), \boldsymbol{\pi}(s, \mathbf{x}(s))) \, ds$$

where the minimum is achieved by the optimal control law $\boldsymbol{\pi}^*(t, \mathbf{x}(t))$.

Suppose that in a given task the system of interest (driven by the optimal control law) generates an average trajectory $\overline{\mathbf{x}}(t)$. On a given trial, let $\Delta\mathbf{x}$ be the deviation form the average trajectory at time $t$. Let $\Delta v^*$ be the change in the optimal cost-to-go $v^*$ due to the deviation $\Delta\mathbf{x}$; i.e., $\Delta v^*(\Delta\mathbf{x}) = v^*(\overline{\mathbf{x}} + \Delta\mathbf{x}) - v^*(\overline{\mathbf{x}})$. Now we are ready to define redundancy: the deviation $\Delta\mathbf{x}$ is redundant iff $\Delta v^*(\Delta\mathbf{x}) = 0$. Note that our definition reduces to the intuitive geometric definition at the end of the movement, where the cost function $\ell$ and optimal cost-to-go $v^*$ are identical.

To define the notion of "correction," we need to separate the passive and active dynamics:

$$\mathbf{f}(t, \mathbf{x}, \mathbf{u}) = \mathbf{a}(t, \mathbf{x}) + B(t, \mathbf{x}) \mathbf{u}$$

The (infinitesimal) expected change in $\mathbf{x}$ due to the control $\mathbf{u} = \boldsymbol{\pi}^*(t, \overline{\mathbf{x}} + \Delta\mathbf{x})$ can now be identified: $\dot{\mathbf{x}}_{\mathbf{u}} = B(t, \overline{\mathbf{x}} + \Delta\mathbf{x}) \boldsymbol{\pi}^*(t, \overline{\mathbf{x}} + \Delta\mathbf{x})$. The corrective action of the control signal is naturally defined as $\mathrm{corr}(\Delta\mathbf{x}) = \langle -\dot{\mathbf{x}}_{\mathbf{u}}, \Delta\mathbf{x} \rangle$.

In order to relate the quantities $\Delta v^*(\Delta\mathbf{x})$ and $\mathrm{corr}(\Delta\mathbf{x})$, we obviously need to know something about the optimal control law $\boldsymbol{\pi}^*$. For problems in the above general form, the optimal control law $\boldsymbol{\pi}^*(t, \mathbf{x}(t))$ is given [7] by the minimum

$$\arg\min_{\mathbf{u}} \; \ell(t, \mathbf{x}, \mathbf{u}) + \mathbf{f}(t, \mathbf{x}, \mathbf{u})^\mathsf{T} v_{\mathbf{x}}^*(t, \mathbf{x}) + \frac{1}{2}\mathrm{trace}\left(G(t, \mathbf{x}, \mathbf{u})^\mathsf{T} v_{\mathbf{xx}}^*(t, \mathbf{x}) G(t, \mathbf{x}, \mathbf{u})\right)$$

where $v_{\mathbf{x}}^*(t, \mathbf{x})$ and $v_{\mathbf{xx}}^*(t, \mathbf{x})$ are the gradient and Hessian of the optimal cost-to-go function $v^*(t, \mathbf{x})$. To be able to minimize this expression explicitly, we will restrict the class of problems to

$$G(t, \mathbf{x}, \mathbf{u}) = [\; C_1(t, \mathbf{x})\mathbf{u} \quad \cdots \quad C_k(t, \mathbf{x})\mathbf{u} \;]$$

$$\ell(t, \mathbf{x}, \mathbf{u}) = q(t, \mathbf{x}) + \frac{1}{2}\mathbf{u}^\mathsf{T} R(t, \mathbf{x})\mathbf{u}$$

The matrix notation means that the $i_{\mathrm{th}}$ column of $G$ is $C_i(t, \mathbf{x})\mathbf{u}$. Note that the latter formulation is still very general, and can represent realistic musculo-skeletal dynamics and motor tasks.

Using the fact[1] that $GG^\mathsf{T} = \sum_{i=1}^{k} C_i \mathbf{u}\mathbf{u}^\mathsf{T} C_i^\mathsf{T}$ and $\operatorname{trace}(UV) = \operatorname{trace}(VU)$, and eliminating terms that do not depend on $\mathbf{u}$, the expression that has to be minimized w.r.t $\mathbf{u}$ becomes

$$\mathbf{u}^\mathsf{T} B\left(t, \mathbf{x}\right)^\mathsf{T} v_{\mathbf{x}}^*\left(t, \mathbf{x}\right) + \frac{1}{2}\mathbf{u}^\mathsf{T} \underbrace{\left(R\left(t, \mathbf{x}\right) + \sum_{i=1}^{k} C_i\left(t, \mathbf{x}\right)^\mathsf{T} v_{\mathbf{xx}}^*\left(t, \mathbf{x}\right) C_i\left(t, \mathbf{x}\right)\right)}_{Z(t, \mathbf{x})}\mathbf{u}$$

Therefore the optimal control law is

$$\boldsymbol{\pi}^*\left(t, \mathbf{x}\right) = -Z\left(t, \mathbf{x}\right)^{-1} B\left(t, \mathbf{x}\right)^\mathsf{T} v_{\mathbf{x}}^*\left(t, \mathbf{x}\right)$$

We now return to the relationship between "redundancy" and "correction." The time index $t$ will be suppressed for clarity. We expand the optimal cost-to-go to second order: $v^*\left(\overline{\mathbf{x}} + \Delta\mathbf{x}\right) \approx v^*\left(\overline{\mathbf{x}} + \Delta\mathbf{x}\right) + \Delta\mathbf{x}^\mathsf{T} v_{\mathbf{x}}^*\left(\overline{\mathbf{x}}\right) + \Delta\mathbf{x}^\mathsf{T} v_{\mathbf{xx}}^*\left(\overline{\mathbf{x}}\right) \Delta\mathbf{x}$, also expand its gradient to first order: $v_{\mathbf{x}}^*\left(\overline{\mathbf{x}} + \Delta\mathbf{x}\right) \approx v_{\mathbf{x}}^*\left(\overline{\mathbf{x}}\right) + v_{\mathbf{xx}}^*\left(\overline{\mathbf{x}}\right) \Delta\mathbf{x}$, and approximate all other quantities as being constant in a small neighborhood of $\overline{\mathbf{x}}$. The effect of the control signal becomes $\dot{\mathbf{x}}_{\mathbf{u}} \approx -B\left(\overline{\mathbf{x}}\right) Z\left(\overline{\mathbf{x}}\right)^{-1} B\left(\overline{\mathbf{x}}\right)^\mathsf{T}\left(v_{\mathbf{x}}^*\left(\overline{\mathbf{x}}\right) + v_{\mathbf{xx}}^*\left(\overline{\mathbf{x}}\right) \Delta\mathbf{x}\right)$. Substituting in the above definitions yields

$$\begin{aligned} \Delta v^*\left(\Delta\mathbf{x}\right) &\approx \left\langle \Delta\mathbf{x}, v_{\mathbf{x}}^*\left(\overline{\mathbf{x}}\right) + v_{\mathbf{xx}}^*\left(\overline{\mathbf{x}}\right) \Delta\mathbf{x}\right\rangle \\ \operatorname{corr}\left(\Delta\mathbf{x}\right) &\approx \left\langle \Delta\mathbf{x}, v_{\mathbf{x}}^*\left(\overline{\mathbf{x}}\right) + v_{\mathbf{xx}}^*\left(\overline{\mathbf{x}}\right) \Delta\mathbf{x}\right\rangle_{B(\overline{\mathbf{x}}) Z(\overline{\mathbf{x}})^{-1} B(\overline{\mathbf{x}})^\mathsf{T}} \end{aligned}$$

where the weighted dot-product notation $\langle \mathbf{x}, \mathbf{y}\rangle_M$ stands for $\mathbf{x}^\mathsf{T} M \mathbf{y}$.

Thus both $\Delta v^*\left(\Delta\mathbf{x}\right)$ and $\operatorname{corr}\left(\Delta\mathbf{x}\right)$ are dot-products of the same two vectors. When $v_{\mathbf{x}}^*\left(\overline{\mathbf{x}}\right) + v_{\mathbf{xx}}^*\left(\overline{\mathbf{x}}\right) \Delta\mathbf{x} = 0$ —which can happen for infinitely many $\Delta\mathbf{x}$ when the Hessian $v_{\mathbf{xx}}^*\left(\overline{\mathbf{x}}\right)$ is singular—the deviation is redundant and the optimal controller takes no corrective action. Furthermore, $\Delta v^*\left(\Delta\mathbf{x}\right)$ and $\operatorname{corr}\left(\Delta\mathbf{x}\right)$ are positively correlated because $B\left(\overline{\mathbf{x}}\right) Z\left(\overline{\mathbf{x}}\right)^{-1} B\left(\overline{\mathbf{x}}\right)^\mathsf{T}$ is a positive semi-definite matrix[2]. Thus the optimal controller resists single-trial deviations that take the system to more costly states, and magnifies deviations to less costly states.

This analysis confirms the minimal intervention principle to be a very general property of optimal feedback controllers, explaining why variability patterns elongated in task-irrelevant dimensions (as well as synergetic actuator coupling) have been observed in such a wide range of experiments involving different actuators and behavioral goals.

## 2.2 Linear-Quadratic-Gaussian (LQG) simulations

The local analysis above is very general, but it leaves a few questions open: i) what happens when the deviation $\Delta\mathbf{x}$ is not small; ii) how does the optimal cost-to-go (which defines redundancy) relate to the cost function (which defines the task); iii) what is the distribution of states resulting from the sequence of optimal control signals? To address such questions (and also build models of specific motor control experiments) we need to focus on a class of control problems for which the optimal control law can actually be found. To that end, we have modified [15] the extensively studied LQG framework to include the multiplicative control noise characteristic of the motor system. The control problems studied here and in

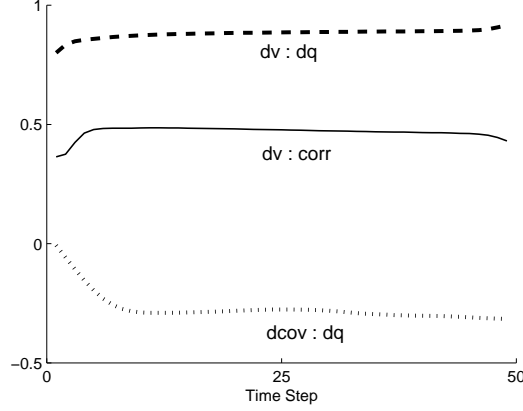

Figure 1: $A, B, C, H, R, Q$ were generated randomly, with the restiction that $A$ has singular values less than 1 (i.e. the passive dynamics is stable); the last component of the state is 1 (for similarity with motor control tasks), $R$ and $Q$ are positive semi-definite, and $Q_t = Q \frac{t}{N}$. For each problem ($N = 50$) and each point in time $t$, we generated 100 random unit vector $\mathbf{d}_i$ and scaled them by mean(sqrt(svd(cov($\mathbf{x}$)))). Then $dv_i \triangleq (\mathbf{d}_i + \overline{\mathbf{x}})^\mathsf{T} S (\mathbf{d}_i + \overline{\mathbf{x}}) - \overline{\mathbf{x}}^\mathsf{T} S \overline{\mathbf{x}}$, $dq_i \triangleq (\mathbf{d}_i + \overline{\mathbf{x}})^\mathsf{T} Q (\mathbf{d}_i + \overline{\mathbf{x}}) - \overline{\mathbf{x}}^\mathsf{T} Q \overline{\mathbf{x}}$, $dcov_i \triangleq (\mathbf{d}_i + \overline{\mathbf{x}})^\mathsf{T} \operatorname{cov}(\mathbf{x}) (\mathbf{d}_i + \overline{\mathbf{x}}) - \overline{\mathbf{x}}^\mathsf{T} \operatorname{cov}(\mathbf{x}) \overline{\mathbf{x}}$, $\operatorname{corr}_i \triangleq \mathbf{d}_i{}^\mathsf{T} B L (\mathbf{d}_i + \overline{\mathbf{x}})$. The notation "dv : dq" stands for the correlation between the $dv_i$ and the $dq_i$, etc.

the next section are in the form

$$
\begin{array}{ll}
\textit{Dynamics} & \mathbf{x}_{t+1} = A\mathbf{x}_t + B\mathbf{u}_t + [\begin{array}{ccc} C_1\mathbf{u}_t & \cdots & C_k\mathbf{u}_t \end{array}] \varepsilon_t \\
\textit{Feedback} & \mathbf{y}_t = H\mathbf{x}_t + \omega_t \\
\textit{Cost} & \mathbf{x}_t^\mathsf{T} Q_t \mathbf{x}_t + \mathbf{u}_t^\mathsf{T} R \mathbf{u}_t
\end{array}
$$

Note that the system state $\mathbf{x}_t$ is now partially observable, through noisy sensor readings $\mathbf{y}_t$. When the noise is additive instead of being multiplicative, the optimal control problem has the well-known solution [5]

$$
\pi_t^*(\widehat{\mathbf{x}}_t) = -L_t\widehat{\mathbf{x}}_t; \quad \widehat{\mathbf{x}}_{t+1} = A\widehat{\mathbf{x}}_t + B\mathbf{u}_t + K_t(\mathbf{y}_t - H\widehat{\mathbf{x}}_t)
$$

where $\widehat{\mathbf{x}}_t$ is an internal estimate of the system state, updated recursively by a Kalman filter. The sequences of matrices $L$ and $K$ are computed from the associated discrete-time Ricatti equations [5]. Multiplicative noise complicates matters, but we have found [15] that for systems with stable passive dynamics a similar control strategy is very close to optimal. The modified equations for $L$ and $K$ are given in [15]. The optimal cost-to-go function is

$$
\begin{aligned}
v_t^*(\widehat{\mathbf{x}}_t) &= \widehat{\mathbf{x}}_t^T S_t \widehat{\mathbf{x}}_t + \text{const} \\
S_t &= Q_t + A^T S_{t+1}(A - BL_t); \quad S_N = Q_N
\end{aligned}
$$

The Hessian $S_t$ of the optimal cost-to-go is closely related to the task cost $Q_t$, but also includes future task costs weighted by the passive ($A$) and closed-loop ($A - BL_t$) dynamics.

Specific motor control tasks are considered below. Here we generate 100 random problems in the above form, compute the optimal control law in each case, and correlate the quantities $\Delta v^*$ and corr. As the "dv : corr" curve in Figure 1 shows, they are positively correlated at all times. We also show in Figure 1 that the Hessian of the optimal cost-to-go has similar shape to the task cost ("dv : dq" curve), and that the state covariance is smaller along dimensions where the task cost is larger; i.e., the correlation "dcov : dq" is negative. See the figure legend for details.

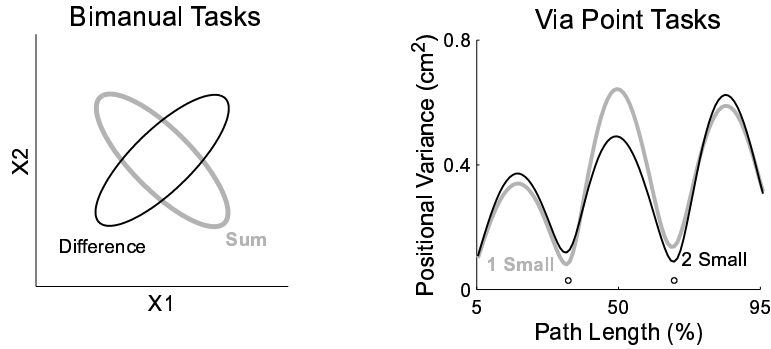

Figure 2: Simulations of motor control tasks – see text.

## 3 Applications to motor coordination

We have used the modified LQG framework to model a wide range of specific motor control tasks [14, 15], and always found that optimal feedback controllers generate variability that is elongated in redundant dimensions. Here we illustrate two such models. The first model (Figure 2, Bimanual Tasks) includes two 1D point masses with positions X1 and X2, each driven with a force actuator whose output is a noisy second-order low-pass filtered version of the corresponding control signal. The feedback contains noisy position, velocity, and force information—delayed by 50 msec (by augmenting the system state with a sequence of recent sensor readings). The " Difference" task requires the two points to start moving 20cm apart, and stop at identical but unspecified locations. The covariance of the final state is elongated in the task-irrelevant dimension: the two points always stop close to each other, but the final location can vary substantially from trial to trial. A related phenomenon has been observed in the more complex bimanual task of inserting a pointer in a cup [6]. We now modify the task: in "Sum," the two points start at the same location and have to stop so that the midpoint between them is at zero. Note that the state covariance is reoriented accordingly. We also illustrate a Via Point task, where a 2D point mass has to pass through a sequence of two intermediate targets and stop at a final target (tracing an S-shaped curve). Variability is minimal at the via points. Furthermore, when one via point is made smaller (i.e., the weight of the corresponding positional constraint is increased), the variability decreases at that point. Due to space limitations, we refer the reader to [14] for details of the models. In [14] we also report a via point experiment that closely matches the predicted effect.

## 4 Multi-attribute costs and desired trajectory tracking

As we stated earlier, replacing the task goal with a desired trajectory (which achieves the goal if executed precisely) is generally suboptimal. A number of examples of such suboptimality are provided in [14]. Here we present a more general view of desired trajectory tracking which clarifies its relationship to optimal control.

Desired trajectory tracking can be incorporated in the present framework by using a modified cost, one that specifies a desired state at each point in time, and penalizes the deviations from that state. Such a modified cost would normally include the original task cost (e.g., the terms that specify the desired terminal state), but also a large number of additional terms that do not need to be minimized in order to accomplish the actual task. This raises the question: what happens to the expected values of the terms in the original cost, when we attempt to minimize other costs simultaneously? Intuitively, one would expect the orig-

inal costs to increase (relative to the costs obtained by the task-optimal controller). The geometric argument below formalizes these ideas, and confirms our intuition.

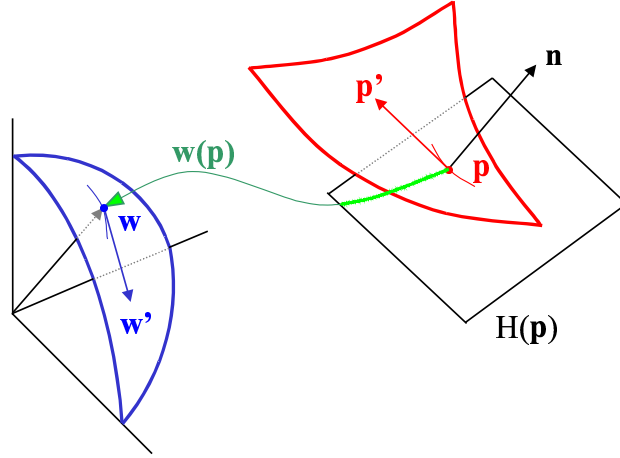

Figure 3

Consider a family of optimal control problems parameterized by the vector $\mathbf{w}$, with cost functions $\ell^{\mathbf{w}}(t, \mathbf{x}, \mathbf{u}) = \sum_{i=1}^{d} w_i \ell_i(t, \mathbf{x}, \mathbf{u})$. Here $\ell_i$ are different component costs, and $w_i$ are the corresponding non-negative weights. Without loss of generality we can assume that $\sum_i w_i^2 = 1$, i.e., the weight vector $\mathbf{w} \in W \subset \mathbb{R}^d$ lies in the positive quadrant of the unit sphere. Let $\boldsymbol{\pi}^{\mathbf{w}}(t, \mathbf{x})$ be an optimal control law[3], and $\mathbf{p}(\mathbf{w}) \in P \subset \mathbb{R}^d$ be the vector of expected component costs achieved by $\boldsymbol{\pi}^{\mathbf{w}}$; i.e.,
$p_i(\mathbf{w}) = E_{\mathbf{x}(.)} \int_0^T \ell_i(t, \mathbf{x}(t), \boldsymbol{\pi}^{\mathbf{w}}(t, \mathbf{x}(t)))\, dt$. Consider a weight vector $\overline{\mathbf{w}}$ and its corresponding $\overline{\mathbf{p}} = \mathbf{p}(\overline{\mathbf{w}})$, such that the mapping $\mathbf{p}(\mathbf{w})$ is locally smooth and invertible. Then we can define the inverse mapping $\mathbf{w}(\mathbf{p})$ from the expected component cost manifold $P$ to the weight manifold $W$, as illustrated in Figure 3.

From the definitions of $\ell^{\mathbf{w}}$ and $p_i$, the total expected cost achieved by $\boldsymbol{\pi}^{\mathbf{w}}$ is $\langle \mathbf{w}(\mathbf{p}), \mathbf{p} \rangle$. Since $\boldsymbol{\pi}^{\mathbf{w}}$ is an optimal control law for the problem defined by the weight vector $\mathbf{w}$, no other control law can achieve a smaller total expected cost, and so $\langle \mathbf{w}(\mathbf{p}), \mathbf{p} \rangle \leq \langle \mathbf{w}(\mathbf{p}), \mathbf{p}^{\#} \rangle$ for all $\mathbf{p}^{\#} \in P$. Therefore, if we construct the $d-1$ dimensional hyperplane $H(\mathbf{p})$ that contains $\mathbf{p}$ and is orthogonal to $\mathbf{w}(\mathbf{p})$, the entire manifold $P$ has to lie in the half-space not containing the origin. Thus $H(\mathbf{p})$ is tangent to the manifold $P$ at point $\mathbf{p}$, $P$ has non-negative curvature, and the unit vector $\mathbf{n}(\mathbf{p})$ which is normal to $P$ at $\mathbf{p}$ satisfies[4] $\mathbf{n}(\mathbf{p}) = \mathbf{w}(\mathbf{p})$.

Let $\mathbf{p}(\alpha) \in P$, $\alpha \in \mathbb{R}$ be a parametric curve that passes through the point of interest $\overline{\mathbf{p}}$: $\mathbf{p}(0) = \overline{\mathbf{p}}$. Define $\mathbf{n}(\alpha) = \mathbf{n}(\mathbf{p}(\alpha))$ and $\mathbf{w}(\alpha) = \mathbf{w}(\mathbf{p}(\alpha))$. By differentiating $\mathbf{p}(\alpha)$ at $\alpha = 0$ we obtain the tangent $\overline{\mathbf{p}}'$ to the curve $\mathbf{p}(\alpha)$ at $\overline{\mathbf{p}}$. Since $\overline{\mathbf{n}}$ is normal to $P$, we have $\langle \overline{\mathbf{n}}, \overline{\mathbf{p}}' \rangle = 0$. Differentiating the latter equality once again yields $\langle \overline{\mathbf{n}}, \overline{\mathbf{p}}'' \rangle + \langle \overline{\mathbf{n}}', \overline{\mathbf{p}}' \rangle = 0$. The non-negative curvature of $P$ implies $\langle \overline{\mathbf{n}}, \overline{\mathbf{p}}'' \rangle \geq 0$; i.e., the tangent $\overline{\mathbf{p}}'$ cannot turn away from the normal $\overline{\mathbf{n}}$ without $\mathbf{p}$ crossing the hyperplane $H$. Therefore $\langle \overline{\mathbf{n}}', \overline{\mathbf{p}}' \rangle \leq 0$, and since $\mathbf{n} = \mathbf{w}$, we have $\langle \overline{\mathbf{w}}', \overline{\mathbf{p}}' \rangle \leq 0$.

The above result means that whenever we change the weight vector $\mathbf{w}$, the corresponding vector $\mathbf{p}(\mathbf{w})$ of expected component costs achieved by the (new) optimal control law will change in an "opposite" direction. More precisely, suppose we vary $\mathbf{w}$ along a great circle that passes through one of the corners of $W$, say $(1, 0, \ldots, 0)$, so that $w_1$ decreases and all $w_{i \neq 1}$ increase. Then the component cost $p_1(\mathbf{w})$ will increase.

## Footnotes

[1] Defining the unit vector $\mathbf{e}_i$ as having a 1 in position $i$ and 0 in all other positions, we can write $G = \sum_{i=1}^{k} C_i \mathbf{u}\mathbf{e}_i^T$. Then $GG^T = \sum_i \sum_j C_i \mathbf{u}\mathbf{e}_i^T \mathbf{e}_j \mathbf{u}^T C_j^T = \sum_i C_i \mathbf{u}\mathbf{u}^T C_i^T$, since $\mathbf{e}_i^T \mathbf{e}_j = \delta_i^j$.

[2] $Z\left(\overline{\mathbf{x}}\right)$ has to be positive semi-definite—or else we could find a control signal that makes the instantaneous cost negative, and that is impossible by definition. Therefore $BZ^{-1}B^\mathsf{T}$ is also positive semi-definite.

[3]If we assume that the optimal control law is unique, all inequalities below become strict.

[4]For a general 2D manifold $P$ embedded in $\mathbb{R}^3$, the mapping $P \to W$ on the unit sphere $W$ that satisfies $\mathbf{n}(\mathbf{p}) = \mathbf{w}(\mathbf{p})$ is known as the Gauss map, and plays an important role in surface differential geometry.

## References

[1] Bernstein, N.I. *The Coordination and Regulation of Movements*. Pergamon Press, (1967).

[2] Bizzi, E., Accornero, N., Chapple, W. & Hogan, N. Posture control and trajectory formation during arm movement. *J Neurosci* 4, 2738-44 (1984).

[3] Cole, K.J. & Abbs, J.H. Kinematic and electromyographic responses to perturbation of a rapid grasp. *J Neurophysiol* 57, 1498-510 (1987).

[4] D'Avella, A. & Bizzi, E. Low dimensionality of supraspinally induced force fields. *PNAS* 95, 7711-7714 (1998).

[5] Davis, M.H.A. & Vinter, R. *Stochastic Modelling and Control*. Chapman and Hall, (1985).

[6] Domkin D., Laczko, J., Jaric, S., Johansson, H., & Latash, M. Structure of joint variability in bimanual pointing tasks. *Exp Brain Res* 143, 11-23 (2002).

[7] Fleming, W. and Soner, H. (1993). *Controlled Markov Processes and Viscosity Solutions*. Applications of Mathematics, Springer-Verlag, Berlin.

[8] Flash, T. & Hogan, N. The coordination of arm movements: an experimentally confirmed mathematical model. *J Neuroscience* 5, 1688-1703 (1985).

[9] Gelfand, I., Gurfinkel, V., Tsetlin, M. & Shik, M. In *Models of the structural-functional organization of certain biological systems*. Gelfand, I., Gurfinkel, V., Fomin, S. & Tsetlin, M. (eds.) MIT Press, 1971.

[10] Harris, C.M. & Wolpert, D.M. Signal-dependent noise determines motor planning. *Nature* 394, 780-784 (1998).

[11] Hinton, G.E. Parallel computations for controlling an arm. *Journal of Motor Behavior* 16, 171-194 (1984).

[12] Robertson, E.M. & Miall, R.C. Multi-joint limbs permit a flexible response to unpredictable events. *Exp Brain Res* 117, 148-52 (1997).

[13] Sutton, G.G. & Sykes, K. The variation of hand tremor with force in healthy subjects. *Journal of Physiology* 191(3), 699-711 (1967).

[14] Todorov, E. & Jordan, M. Optimal feedback control as a theory of motor coordination. *Nature Neuroscience*, 5(11), 1226-1235 (2002).

[15] Todorov, E. Optimal feedback control under signal-dependent noise: Methodology for modeling biological movement. *Neural Computation*, under review. Available at http://cogsci.ucsd.edu/~todorov. (2002).

[16] Scholz, J.P. & Schoner, G. The uncontrolled manifold concept: Identifying control variables for a functional task. *Exp Brain Res* 126, 289-306 (1999).

[17] Uno, Y., Kawato, M. & Suzuki, R. Formation and control of optimal trajectory in human multijoint arm movement: Minimum torque-change model. *Biological Cybernetics* 61, 89-101 (1989).

[18] Santello, M. & Soechting, J.F. Force synergies for multifingered grasping. *Exp Brain Res* 133, 457-67 (2000).
